# HM-BiTAM: Bilingual Topic Exploration, Word Alignment, and Translation

**Bing Zhao**
IBM T. J. Watson Research
zhaob@us.ibm.com

**Eric P. Xing**
Carnegie Mellon University
epxing@cs.cmu.edu

## Abstract

We present a novel paradigm for statistical machine translation (SMT), based on a joint modeling of word alignment and the topical aspects underlying bilingual document-pairs, via a hidden Markov Bilingual Topic AdMixture (HM-BiTAM). In this paradigm, parallel sentence-pairs from a parallel document-pair are coupled via a certain semantic-flow, to ensure coherence of topical context in the alignment of mapping words between languages, likelihood-based training of topic-dependent translational lexicons, as well as in the inference of topic representations in each language. The learned HM-BiTAM can not only display topic patterns like methods such as LDA [1], but now for bilingual corpora; it also offers a principled way of inferring optimal translation using document context. Our method integrates the conventional model of HMM — a key component for most of the state-of-the-art SMT systems, with the recently proposed BiTAM model [10]; we report an extensive empirical analysis (in many ways complementary to the description-oriented [10]) of our method in three aspects: bilingual topic representation, word alignment, and translation.

## 1 Introduction

Most contemporary SMT systems view parallel data as independent sentence-pairs whether or not they are from the same document-pair. Consequently, translation models are learned only at sentence-pair level, and document contexts – essential factors for translating documents – are generally overlooked. Indeed, translating documents differs considerably from translating a group of unrelated sentences. A sentence, when taken out of the context from the document, is generally more ambiguous and less informative for translation. One should avoid destroying a coherent document by simply translating it into a group of sentences which are indifferent to each other and detached from the context.

Developments in statistics, genetics, and machine learning have shown that latent semantic aspects of complex data can often be captured by a model known as the *statistical admixture* (or mixed membership model [4]). Statistically, an object is said to be derived from an admixture if it consists of a bag of elements, each sampled independently or coupled in a certain way, from a mixture model. In the context of SMT, each parallel document-pair is treated as one such object. Depending on the chosen modeling granularity, all sentence-pairs or word-pairs in a document-pair correspond to the basic elements constituting the object, and the mixture from which the elements are sampled can correspond to a collection of translation lexicons and monolingual word frequencies based on different topics (e.g., economics, politics, sports, etc.). Variants of admixture models have appeared in population genetics [6] and text modeling [1, 4].

Recently, a *Bilingual Topic-AdMixture* (**BiTAM**) model was proposed to capture the topical aspects of SMT [10]; word-pairs from a parallel document-pair follow the same weighted mixtures of translation lexicons, inferred for the given document-context. The BiTAMs generalize over IBM Model-1; they are efficient to learn and scalable for large training data. However, they do not capture locality

constraints of word alignment, i.e., words "close-in-source" are usually aligned to words "close-in-target", under document-specific topical assignment. To incorporate such constituents, we integrate the strengths of both HMM and BiTAM, and propose a Hidden Markov Bilingual Topic-AdMixture model, or HM-BiTAM, for word alignment to leverage both locality constraints and topical context underlying parallel document-pairs.

In the HM-BiTAM framework, one can estimate topic-specific word-to-word translation lexicons (lexical mappings), as well as the monolingual topic-specific word-frequencies for both languages, based on parallel document-pairs. The resulting model offers a principled way of inferring optimal translation from a given source language in a context-dependent fashion. We report an extensive empirical analysis of HM-BiTAM, in comparison with related methods. We show our model's effectiveness on the word-alignment task; we also demonstrate two application aspects which were *untouched* in [10]: the utility of HM-BiTAM for bilingual topic exploration, and its application for improving translation qualities.

## 2 Revisit HMM for SMT

An SMT system can be formulated as a noisy-channel model [2]:

$$e^* = \arg\max_e P(e|f) = \arg\max_e P(f|e)P(e), \tag{1}$$

where a translation corresponds to searching for the *target* sentence $e^*$ which explains the *source* sentence $f$ best. The key component is $P(f|e)$, the translation model; $P(e)$ is monolingual language model. In this paper, we generalize $P(f|e)$ with topic-admixture models.

An HMM implements the "proximity-bias" assumption — that words "close-in-source" are aligned to words "close-in-target", which is effective for improving word alignment accuracies, especially for linguistically close language-pairs [8]. Following [8], to model word-to-word translation, we introduce the mapping $j \rightarrow a_j$, which assigns a French word $f_j$ in position $j$ to an English word $e_i$ in position $i = a_j$ denoted as $e_{a_j}$. Each (ordered) French word $f_j$ is an observation, and it is generated by an HMM state defined as $[e_{a_j}, a_j]$, where the alignment indicator $a_j$ for position $j$ is considered to have a dependency on the previous alignment $a_{j-1}$. Thus a first-order HMM for an alignment between $e \equiv e_{1:I}$ and $f \equiv f_{1:J}$ is defined as:

$$p(f_{1:J}|e_{1:I}) = \sum_{a_{1:J}} \prod_{j=1}^{J} p(f_j|e_{a_j})p(a_j|a_{j-1}), \tag{2}$$

where $p(a_j|a_{j-1})$ is the *state transition probability*; $J$ and $I$ are *sentence lengths* of the French and English sentences, respectively. The transition model enforces the proximity-bias. An additional pseudo word "NULL" is used at the beginning of English sentences for HMM to start with. The HMM implemented in GIZA++ [5] is used as our baseline, which includes refinements such as special treatment of a jump to a NULL word. A graphical model representation for such an HMM is illustrated in Figure 1 (a).

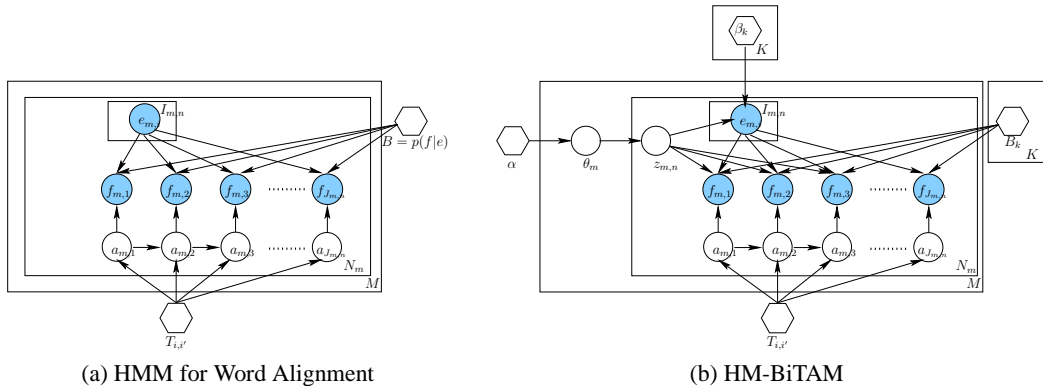

(a) HMM for Word Alignment          (b) HM-BiTAM

Figure 1: The graphical model representations of (a) HMM, and (b) HM-BiTAM, for parallel corpora. Circles represent random variables, hexagons denote parameters, and observed variables are shaded.

# 3 Hidden Markov Bilingual Topic-AdMixture

We assume that in training corpora of bilingual documents, the *document-pair boundaries are known*, and indeed they serve as the key information for defining document-specific topic weights underlying aligned *sentence-pairs* or *word-pairs*. To simplify the outline, the topics here are sampled at sentence-pair level; topics sampled at word-pair level can be easily derived following the outlined algorithms, in the same spirit of [10]. Given a document-pair $(\mathbf{F}, \mathbf{E})$ containing $N$ parallel sentence-pairs $(\mathbf{e}_n, \mathbf{f}_n)$, HM-BiTAM implements the following generative scheme.

## 3.1 Generative Scheme of HM-BiTAM

Given a conjugate prior Dirichlet($\alpha$), the *topic-weight vector* (hereafter, TWV), $\theta_m$ for each document-pair $(\mathbf{F}_m, \mathbf{E}_m)$, is sampled independently. Let the non-underscripted $\theta$ denote the TWV of a typical document-pair $(\mathbf{F}, \mathbf{E})$, a collection of topic-specific translation lexicons be $\mathbf{B} \equiv \{B_k\}$, where $\mathbf{B}_{i,j,k} = P(f=f_j|e=e_i, z=k)$ is the conditional probability of translating $e$ into $f$ under a given topic indexed by $z$; the topic-specific monolingual model $\beta \equiv \{\beta_k\}$, which can be the usual LDA-style monolingual unigrams. The sentence-pairs $\{\mathbf{f}_n, \mathbf{e}_n\}$ are drawn independently from a mixture of topics. Specifically (as illustrated also in Fig. 1 (b)):

1. $\theta \sim \text{Dirichlet}(\alpha)$

2. For each sentence-pair $(\mathbf{f}_n, \mathbf{e}_n)$,
   (a) $z_n \sim \text{Multinomial}(\theta)$     sample the topic
   (b) $e_{n,1:I_n}|z_n \sim P(\mathbf{e}_n|z_n; \beta)$     sample all English words from a monolingual topic model (e.g., an unigram model),
   (c) For each position $j_n = 1, \ldots, J_n$ in $\mathbf{f}_n$,
      i. $a_{j_n} \sim P(a_{j_n}|a_{j_n-1}; T)$     sample an alignment link $a_{j_n}$ from a first-order Markov process,
      ii. $f_{j_n} \sim P(f_{j_n}|\mathbf{e}_n, a_{j_n}, z_n; \mathbf{B})$     sample a foreign word $f_{j_n}$ according to a topic specific translation lexicon.

Under an HM-BiTAM model, each sentence-pair consists of a mixture of latent bilingual topics; each topic is associated with a distribution over bilingual word-pairs. Each word $f$ is generated by two hidden factors: a latent topic $z$ drawn from a document-specific distribution over $K$ topics, and the English word $e$ identified by the hidden alignment variable $a$.

## 3.2 Extracting Bilingual Topics from HM-BiTAM

Because of the *parallel nature* of the data, the topics of English and the foreign language will share similar semantic meanings. This assumption is captured in our model. Shown in Figure 1(b), both the English and foreign topics are sampled from the same distribution $\theta$, which is a document-specific topic-weight vector.

Although there is an inherent asymmetry in the bilingual topic representation in HM-BiTAM (that the monolingual topic representations $\beta$ are only defined for English, and the foreign topic representations are implicit via the topical translation models), it is not difficult to retrieve the monolingual topic representations of the foreign language via a marginalization over hidden word alignment. For example, the frequency (i.e., unigram) of foreign word $f_w$ under topic $k$ can be computed by

$$P(f_w|k) = \sum_e P(f_w|e, B_k)P(e|\beta_k). \tag{3}$$

As a result, HM-BiTAM can actually be used as a bilingual topic explorer in the LDA-style and beyond. Given paired documents, it can extract the representations of each topic in both languages in a consistent fashion (which is not guaranteed if topics are extracted separately from each language using, e.g., LDA), as well as the lexical mappings under each topics, based on a maximal likelihood or Bayesian principle. In Section 5.2, we demonstrate outcomes of this application.

We expect that, under the HM-BiTAM model, because bilingual statistics from word alignment $a$ are shared effectively across different topics, a word will have much less translation candidates due to constraints by the hidden topics; therefore the topic specific translation lexicons are much *smaller* and *sharper*, which give rise to a more parsimonious and unambiguous translation model.

# 4 Learning and Inference

We sketch a generalized mean-field approximation scheme for inferring latent variables in HM-BiTAM, and a variational EM algorithm for estimating model parameters.

## 4.1 Variational Inference

Under HM-BiTAM, the complete likelihood of a document-pair $(\mathbf{F}, \mathbf{E})$ can be expressed as follows:

$$p(\mathbf{F}, \mathbf{E}, \theta, \vec{z}, \vec{a} | \alpha, \beta, T, \mathbf{B}) = p(\theta|\alpha) P(\vec{z}|\theta) P(\vec{a}|T) P(\mathbf{F}|\vec{a}, \vec{z}, \mathbf{E}, \mathbf{B}) P(\mathbf{E}|\vec{z}, \beta), \tag{4}$$

where $P(\vec{a}|T) = \prod_{n=1}^{N} \prod_{j=1}^{J_n} P(a_{j_n}|a_{j_n-1}; T)$ represents the probability of a sequence of alignment jumps; $P(\mathbf{F}|\vec{a}, \vec{z}, \mathbf{E}, \mathbf{B}) = \prod_{n=1}^{N} \prod_{j=1}^{J_n} P(f_{j_n}|a_{j_n}, \mathbf{e}_n, z_n, \mathbf{B})$ is the *document-level* translation probability; and $P(\mathbf{E}|\vec{z}, \beta)$ is the topic-conditional likelihood of the English document based on a topic-dependent unigram as used in LDA. Apparently, exact inference under this model is infeasible as noted in earlier models related to, but simpler than, this one [10].

To approximate the posterior $p(\vec{a}, \theta, \vec{z}|\mathbf{F}, \mathbf{E})$, we employ a generalized mean field approach and adopt the following factored approximation to the true posterior: $q(\theta, \vec{z}, \vec{a}) = q(\theta|\vec{\gamma}) q(\vec{z}|\vec{\phi}) q(\vec{a}|\vec{\lambda})$, where $q(\theta|\vec{\gamma})$, $q(\vec{z}|\vec{\phi})$, and $q(\vec{a}|\vec{\lambda})$ are re-parameterized Dirichlet, multinomial, and HMM, respectively, determined by some *variational parameters* that correspond to the expected sufficient statistics of the dependent variables of each factor [9].

As well known in the variational inference literature, solutions to the above variational parameters can be obtained by minimizing the Kullback-Leibler divergence between $q(\theta, \vec{z}, \vec{a})$ and $p(\theta, \vec{z}, \vec{a}|\mathbf{F}, \mathbf{E})$, or equivalently, by optimizing the lower-bound of the expected (over $q()$) log-likelihood defined by Eq.(4), via a fixed-point iteration. Due to space limit, we forego a detailed derivation, and directly give the fixed-point equations below:

$$\hat{\gamma}_k = \alpha_k + \sum_{n=1}^{N} \phi_{n,k}, \tag{5}$$

$$\hat{\phi}_{n,k} \propto \exp\Big(\Psi(\gamma_k) - \Psi(\sum_{k=1}^{K} \gamma_k)\Big) \cdot \exp\Big(\sum_{i=1}^{I_n}\sum_{j=1}^{J_n} \lambda_{n,j,i} \log \beta_{k,e_{i_n}}\Big)$$

$$\times \exp\Big(\sum_{j,i=1}^{J_n,I_n}\sum_{f\in V_F}\sum_{e\in V_E} \mathbf{1}(f_{j_n},f)\mathbf{1}(e_{i_n},e)\lambda_{n,j,i}\log B_{f,e,k}\Big), \tag{6}$$

$$\hat{\lambda}_{n,j,i} \propto \exp\Big(\sum_{i'=1}^{I_n} \lambda_{n,j-1,i'} \log T_{i,i'}\Big) \times \exp\Big(\sum_{i''=1}^{I_n} \lambda_{n,j+1,i''} \log T_{i'',i}\Big)$$

$$\times \exp\Big(\sum_{f\in V_F}\sum_{e\in V_E}\mathbf{1}(f_{j_n},f)\mathbf{1}(e_{i_n},e)\sum_{k=1}^{K}\phi_{n,k}\log B_{f,e,k}\Big) \times \exp\Big(\sum_{k=1}^{K}\phi_{n,k}\log \beta_{k,e_{i_n}}\Big), \tag{7}$$

where $\mathbf{1}(\cdot,\cdot)$ denotes an indicator function, and $\Psi(\cdot)$ represents the digamma function.

The vector $\hat{\phi}_n \equiv (\hat{\phi}_{n,1},\ldots,\hat{\phi}_{n,K})$ given by Eq. (6) represents the approximate posterior of the topic weights for each sentence-pair $(\mathbf{f}_n, \mathbf{e}_n)$. The topical information for updating $\hat{\phi}_n$ is collected from three aspects: aligned word-pairs weighted by the corresponding topic-specific translation lexicon probabilities, topical distributions of monolingual English language model, and the smoothing factors from the topic prior.

Equation (7) gives the approximate posterior probability for alignment between the $j$-th word in $\mathbf{f}_n$ and the $i$-th word in $\mathbf{e}_n$, in the form of an exponential model. Intuitively, the first two terms represent the messages corresponding to the *forward* and the *backward* passes in HMM; The third term represents the *emission* probabilities, and it can be viewed as a geometric interpolation of the strengths of individual topic-specific lexicons; and the last term provides further smoothing from monolingual topic-specific aspects.

**Inference of optimum word-alignment** One of the translation model's goals is to infer the optimum word alignment: $a^* = \arg\max_a P(a|\mathbf{F}, \mathbf{E})$. The variational inference scheme described above leads to an *approximate* alignment posterior $q(\vec{a}|\vec{\lambda})$, which is in fact a reparameterized HMM. Thus, extracting the optimum alignment amounts to applying an Viterbi algorithm on $q(\vec{a}|\vec{\lambda})$.

## 4.2 Variational EM for parameter estimation

To estimate the HM-BiTAM parameters, which include the Dirichlet hyperparameter $\alpha$, the transition matrix $T$, the topic-specific monolingual English unigram $\{\vec{\beta}_k\}$, and the topic-specific translation lexicon $\{B_k\}$, we employ an variational EM algorithm which iterates between computing variational distribution of the hidden variables (the **E-step**) as described in the previous subsection, and optimizing the parameters with respect to the variational likelihood (the **M-step**). Here are the update equations for the M-step:

$$\hat{T}_{i'',i'} \propto \sum_{n=1}^{N} \sum_{j=1}^{J_n} \lambda_{n,j,i''} \lambda_{n,j-1,i'}, \tag{8}$$

$$B_{f,e,k} \propto \sum_{n=1}^{N} \sum_{j=1}^{J_n} \sum_{i=1}^{I_n} \sum_{k=1}^{K} \mathbf{1}(f_{j_n}, f)\mathbf{1}(e_{i_n}, e)\lambda_{n,j,i}\phi_{n,k}, \tag{9}$$

$$\beta_{k,e} \propto \sum_{n=1}^{N} \sum_{i=1}^{I_n} \sum_{j=1}^{J_n} 1_{e_i,e}\lambda_{nji}\phi_{n,k}. \tag{10}$$

For updating Dirichlet hyperparameter $\alpha$, which is a corpora-level parameter, we resort to gradient accent as in [7]. The overall computation complexity of the model is linear to the number of topics.

## 5 Experiments

In this section, we investigate three main aspects of the HM-BiTAM model, including word alignment, bilingual topic exploration, and machine translation.

| Train | #Doc. | #Sent. | #Tokens | |
|---|---|---|---|---|
| | | | English | Chinese |
| TreeBank | 316 | 4172 | 133,598 | 105,331 |
| Sinorama04 | 6367 | 282176 | 10,321,061 | 10,027,095 |
| Sinorama02 | 2373 | 103252 | 3,810,664 | 3,146,014 |
| Chnews.2005 | 1001 | 10317 | 326,347 | 270,274 |
| FBIS.BEIJING | 6111 | 99396 | 4,199,030 | 3,527,786 |
| XinHua.NewsStory | 17260 | 98444 | 3,807,884 | 3,915,267 |
| ALL | 33,428 | 597,757 | 22,598,584 | 20,991,767 |

Table 1: Training data statistics.

The training data is a collection of parallel *document-pairs*, with document boundaries explicitly given. As shown in Table 1, our training corpora are general newswire, covering topics mainly about *economics*, *politics*, *educations* and *sports*. For word-alignment evaluation, our test set consists of *95* document-pairs, with *627* manually-aligned sentence-pairs and *14,769* alignment-links in total, from TIDES'01 dryrun data. Word segmentations and tokenizations were fixed manually for optimal word-alignment decisions. This test set contains relatively long sentence-pairs, with an average sentence length of *40.67* words. The long sentences introduce more ambiguities for alignment tasks.

For testing translation quality, TIDES'02 MT evaluation data is used as development data, and ten documents from TIDES'04 MT-evaluation are used as the unseen test data. BLEU scores are reported to evaluate translation quality with HM-BiTAM models.

### 5.1 Empirical Validation

**Word Alignment Accuracy** We trained HM-BiATMs with ten topics using parallel corpora of sizes ranging from 6M to 22.6M words; we used the F-measure, the harmonic mean of precision and recall, to evaluate word-alignment accuracy. Following the same logics for all BiTAMs in [10], we choose HM-BiTAM in which topics are sampled at word-pair level over sentence-pair level. The baseline IBM models were trained using a $1^8 h^5 4^3$ scheme [2]. Refined alignments are obtained from both directions of baseline models in the same way as described in [5].

Figure 2 shows the alignment accuracies of HM-BiTAM, in comparison with that of the baseline-HMM, the baseline BiTAM, and the IBM Model-4. Overall, HM-BiTAM gives significantly better F-measures over HMM, with absolute margins of 7.56%, 5.72% and 6.91% on training sizes of

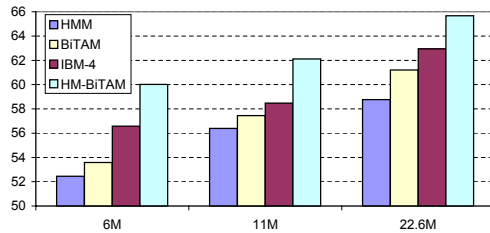

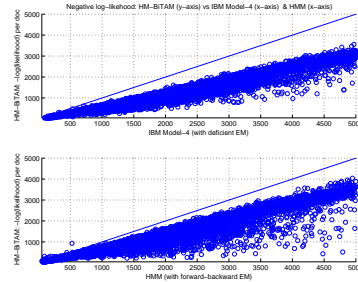

Figure 2: Alignment accuracy (F-measure) of different models trained on corpora of different sizes.

Figure 3: Comparison of likelihoods of data under different models. Top: HM-BiTAM v.s. IBM Model-4; bottom: HM-BiTAM v.s. HMM.

6 M, 11 M and 22.6 M words, respectively. In HM-BiTAM, two factors contribute to narrowing down the word-alignment decisions: the position and the lexical mapping. The position part is the same as the baseline-HMM, implementing the "proximity-bias". Whereas the emission lexical probability is different, each state is a mixture of topic-specific translation lexicons, of which the weights are inferred using document contexts. The topic-specific translation lexicons are sharper and smaller than the global one used in HMM. Thus the improvements of HM-BiTAM over HMM essentially resulted from the extended topic-admixture lexicons. Not surprisingly, HM-BiTAM also outperforms the baseline-BiTAM significantly, because BiTAM captures only the topical aspects and ignores the proximity bias.

Notably, HM-BiTAM also outperforms IBM Model-4 by a margin of 3.43%, 3.64% and 2.73%, respectively. Overall, with 22.6 M words, HM-BiTAM outperforms HMM, BiTAM, IBM-4 significantly, $p$=0.0031, 0.0079, 0.0121, respectively. IBM Model-4 already integrates the fertility and distortion submodels on top of HMM, which further narrows the word-alignment choices. However, IBM Model-4 does not have a scheme to adjust its lexicon probabilities specific to document topical-context as in HM-BiTAM. In a way, HM-BiTAM wins over IBM-4 by leveraging topic models that capture the document context.

**Likelihood on Training and Unseen Documents**   Figure 3 shows comparisons of the likelihoods of document-pairs in the training set under HM-BiTAM with those under IBM Model-4 or HMM. Each point in the figure represents one document-pair; the $y$-coordinate corresponds to the negative log-likelihood under HM-BiTAM, and the $x$-coordinate gives the counterparts under IBM Model-4 or HMM. Overall the likelihoods under HM-BiTAM are significantly better than those under HMM and IBM Model-4, revealing the better modeling power of HM-BiTAM.

We also applied HM-BiTAM to ten document-pairs selected from MT04, which were not included in the training. These document-pairs contain long sentences and diverse topics. As shown in Table 2, the likelihoods of HM-BiTAM on these unseen data dominates significantly over that of HMM, BiTAM, and IBM Models in every case, confirming that HM-BiTAM indeed offers a better fit and generalizability for the bilingual document-pairs.

| Publishers | Genre | IBM-1 | HMM | IBM-4 | BiTAM | HM-BiTAM |
|---|---|---|---|---|---|---|
| AgenceFrance(AFP) | news | -3752.94 | -3388.72 | -3448.28 | -3602.28 | -3188.90 |
| AgenceFrance(AFP) | news | -3341.69 | -2899.93 | -3005.80 | -3139.95 | -2595.72 |
| AgenceFrance(AFP) | news | -2527.32 | -2124.75 | -2161.31 | -2323.11 | -2063.69 |
| ForeignMinistryPRC | speech | -2313.28 | -1913.29 | -1963.24 | -2144.12 | -1669.22 |
| HongKongNews | speech | -2198.13 | -1822.25 | -1890.81 | -2035 | -1423.84 |
| People's Daily | editorial | -2485.08 | -2094.90 | -2184.23 | -2377.1 | -1867.13 |
| United Nation | speech | -2134.34 | -1755.11 | -1821.29 | -1949.39 | -1431.16 |
| XinHua News | news | -2425.09 | -2030.57 | -2114.39 | -2192.9 | -1991.31 |
| XinHua News | news | -2684.85 | -2326.39 | -2352.62 | -2527.78 | -2317.47 |
| ZaoBao News | editorial | -2376.12 | -2047.55 | -2116.42 | -2235.79 | -1943.25 |
| Avg. Perplexity | | 123.83 | 60.54 | 68.41 | 107.57 | 43.71 |

Table 2: Likelihoods of unseen documents under HM-BiTAMs, in comparison with competing models.

## 5.2   Application 1: Bilingual Topic Extraction

**Monolingual topics:**   HM-BiTAM facilitates inference of the latent LDA-style representations of topics [1] in both English and the foreign language (i.e., Chinese) from a given bilingual corpora. The English topics (represented by the topic-specific word frequencies) can be directly read-off from HM-BiTAM parameters $\beta$. As discussed in § 3.2, even though the topic-specific distributions

of words in the Chinese corpora are not directly encoded in HM-BiTAM, one can marginalize over alignments of the parallel data to synthesize them based on the monolingual English topics and the topic-specific lexical mapping from English to Chinese.

Figure 4 shows five topics, in both English and Chinese, learned via HM-BiTAM. The top-ranked frequent words in each topic exhibit coherent semantic meanings; and there are also consistencies between the word semantics under the same topic indexes across languages. Under HM-BiTAM, the two respective monolingual word-distributions for the same topic are statistically coupled due to sharing of the same topic for each sentence-pair in the two languages. Whereas if one merely apply LDA to the corpora in each language separately, such coupling can not be exploited. This coupling enforces consistency between the topics across languages. However, like general clustering algorithms, topics in HM-BiTAM, are not necessarily to present obvious semantic labels.

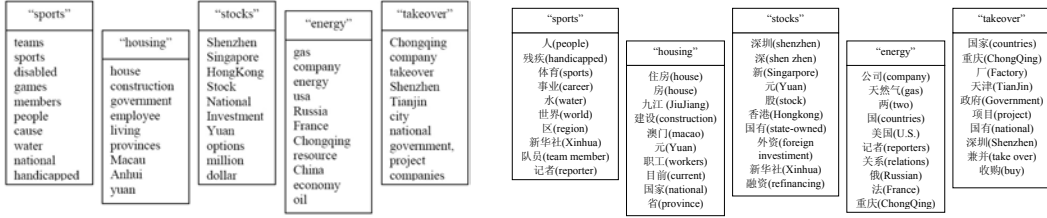

Figure 4: Monolingual topics of both languages learned from parallel data. It appears that the English topics (on the left panel) are highly parallel to the Chinese ones (annotated with English gloss, on the right panel).

**Topic-Specific Lexicon Mapping:**  Table 3 shows two examples of topic-specific lexicon mapping learned by HM-BiTAM. Given a topic assignment, a word usually has much less translation candidates, and the topic-specific translation lexicons are generally much smaller and sharper. Different topic-specific lexicons emphasize different aspects of translating the same source words, which can not be captured by the IBM models or HMM. This effect can be observed from Table 3.

| Topics | "meet" | | | "power" | | |
| --- | --- | --- | --- | --- | --- | --- |
| | TopCand | Meaning | Probability | TopCand | Meaning | Probability |
| Topic-1 | 运动会 | sports meeting | 0.508544 | 电力 | electric power | 0.565666 |
| Topic-2 | 满足 | to satisfy | 0.160218 | 电厂 | electricity factory | 0.656 |
| Topic-3 | 适应 | to adapt | 0.921168 | 涉及 | to be relevant | 0.985341 |
| Topic-4 | 调整 | to adjust | 0.996929 | 力量 | strength | 0.410503 |
| Topic-5 | 会见 | to see someone | 0.693673 | 力量 | strength | 0.997586 |
| Topic-6 | - | - | - | - | - | - |
| Topic-7 | 满足 | to satisfy | 0.467555 | 瓦 | Electric watt | 0.613711 |
| Topic-8 | 运动会 | sports meeting | 0.487728 | 实力 | power | 1.0 |
| Topic-9 | - | - | - | 输 | to generate | 0.50457 |
| Topic-10 | 会见 | to see someone | 0.551466 | 力量 | strength | 1.0 |
| IBM Model-1 | 运动会 | sports meeting | 0.590271 | 电厂 | power plant | 0.314349 |
| HMM | 运动会 | sports meeting | 0.72204 | 力量 | strength | 0.51491 |
| IBM Model-4 | 运动会 | sports meeting | 0.608391 | 力量 | strength | 0.506258 |

Table 3: Topic-specific translation lexicons learned by HM-BiTAM. We show the top candidate (TopCand) lexicon mappings of "meet" and "power" under ten topics. (The symbol "-" means inexistence of significant lexicon mapping under that topic.) Also shown are the semantic meanings of the mapped Chinese words, and the mapping probability $p(f|e, k)$.

## 5.3  Application 2: Machine Translation

The *parallelism* of topic-assignment between languages modeled by HM-BiTAM, as shown in § 3.2 and exemplified in Fig. 4, enables a natural way of improving translation by exploiting semantic consistency and contextual coherency more explicitly and aggressively. Under HM-BiTAM, given a source document $D_F$, the predictive probability distribution of candidate translations of every source word, $P(e|f, D_F)$, must be computed by mixing multiple topic-specific translation lexicons according to the topic weights $p(z|D_F)$ determined from monolingual context in $D_F$. That is:

$$P(e|f, D_F) \propto P(f|e, D_F)P(e|D_F) = \sum_{k=1}^{K} P(f|e, z = k)P(e|z = k)P(z = k|D_F). \qquad (11)$$

We used $p(e|f, D_F)$ to score the bilingual phrase-pairs in a state-of-the-art GALE translation system trained with 250 M words. We kept all other parameters the same as those used in the baseline. Then decoding of the unseen ten MT04 documents in Table 2 was carried out.

| Systems | 1-gram | 2-gram | 3-gram | 4-gram | BLEUr4n4 |
|---|---|---|---|---|---|
| Hiero Sys. | 73.92 | 40.57 | 23.21 | 13.84 | 30.70 |
| Gale Sys. | 75.63 | 42.71 | 25.00 | 14.30 | **32.78** |
| HM-BiTAM | **76.77** | **42.99** | **25.42** | **14.56** | 33.19 |
| Ground Truth | **76.10** | **43.85** | **26.70** | **15.73** | 34.17 |

Table 4: Decoding MT04 10-documents. Experiments using the topic assignments inferred from ground truth and the ones inferred via HM-BITAM; ngram precisions together with final BLEUr4n4 scores are evaluated.

Table 4 shows the performance of our in-house Hiero system (following [3]), the state-of-the-art Gale-baseline (with a better BLEU score), and our HM-BiTAM model, on the NIST MT04 test set. If we know the ground truth of translation to infer the topic-weights, improvement is from $32.78$ to $34.17$ BLEU points. With topical inference from HM-BiTAM using monolingual source document, improved N-gram precisions in the translation were observed from 1-gram to 4-gram. The largest improved precision is for unigram: from $75.63\%$ to $76.77\%$. Intuitively, unigrams have potentially more ambiguities for translations than the higher order ngrams, because the later ones encode already contextual information. The overall BLEU score improvement of HM-BiTAM over other systems, including the state-of-the-art, is from $32.78$ to $33.19$, an slight improvement with $p = 0.043$.

## 6 Discussion and Conclusion

We presented a novel framework, HM-BiTAM, for exploring bilingual topics, and generalizing over traditional HMM for improved word-alignment accuracies and translation quality. A variational inference and learning procedure was developed for efficient training and application in translation. We demonstrated significant improvement of word-alignment accuracy over a number of existing systems, and the interesting capability of HM-BiTAM to simultaneously extract coherent monolingual topics from both languages. We also report encouraging improvement of translation quality over current benchmarks; although the margin is modest, it is noteworthy that the current version of HM-BiTAM remains a purely autonomously trained system. Future work also includes extensions with more structures for word-alignment such as noun phrase chunking.

## Footnotes

[2]Eight iterations for IBM Model-1, five iterations for HMM, and three iterations for IBM Model-4 (with deficient EM: normalization factor is computed using sampled alignment neighborhood in E-step)

## References

[1] David Blei, Andrew NG, and Michael I. Jordon. Latent dirichlet allocation. In *Journal of Machine Learning Research*, volume 3, pages 1107–1135, 2003.

[2] Peter F. Brown, Stephen A. Della Pietra, Vincent. J. Della Pietra, and Robert L. Mercer. The mathematics of statistical machine translation: Parameter estimation. In *Computational Linguistics*, volume 19(2), pages 263–331, 1993.

[3] David Chiang. A hierarchical phrase-based model for statistical machine translation. In *Proceedings of the 43rd Annual Meeting of the Association for Computational Linguistics (ACL'05)*, pages 263–270, Ann Arbor, Michigan, June 2005. Association for Computational Linguistics.

[4] Elena Erosheva, Steve Fienberg, and John Lafferty. Mixed membership models of scientific publications. In *Proceedings of the National Academy of Sciences*, volume 101 of *Suppl. 1*, April 6 2004.

[5] Franz J. Och and Hermann Ney. The alignment template approach to statistical machine translation. In *Computational Linguistics*, volume 30, pages 417–449, 2004.

[6] J. Pritchard, M. Stephens, and P. Donnell. Inference of population structure using multilocus genotype data. In *Genetics*, volume 155, pages 945–959, 2000.

[7] K. Sjölander, K. Karplus, M. Brown, R. Hughey, A. Krogh, I.S. Mian, and D. Haussler. Dirichlet mixtures: A method for improving detection of weak but significant protein sequence homology. *Computer Applications in the Biosciences*, 12, 1996.

[8] Stephan. Vogel, Hermann Ney, and Christoph Tillmann. HMM based word alignment in statistical machine translation. In *Proc. The 16th Int. Conf. on Computational Lingustics, (Coling'96)*, pages 836–841, Copenhagen, Denmark, 1996.

[9] Eric P. Xing, M.I. Jordan, and S. Russell. A generalized mean field algorithm for variational inference in exponential families. In Meek and Kjaelff, editors, *Uncertainty in Artificial Intelligence (UAI2003)*, pages 583–591. Morgan Kaufmann Publishers, 2003.

[10] Bing Zhao and Eric P. Xing. Bitam: Bilingual topic admixture models for word alignment. In *Proceedings of the 44th Annual Meeting of the Association for Computational Linguistics (ACL'06)*, 2006.

